# The Devil and the Network: What Sparsity Implies to Robustness and Memory

**Sanjay Biswas** and **Santosh S. Venkatesh**
Department of Electrical Engineering
University of Pennsylvania
Philadelphia, PA 19104

## Abstract

Robustness is a commonly bruited property of neural networks; in particular, a folk theorem in neural computation asserts that neural networks—in contexts with large interconnectivity—continue to function efficiently, albeit with some degradation, in the presence of component damage or loss. A second folk theorem in such contexts asserts that dense interconnectivity between neural elements is a *sine qua non* for the efficient usage of resources. These premises are formally examined in this communication in a setting that invokes the notion of the "devil"[1] in the network as an agent that produces sparsity by snipping connections.

## 1 ON REMOVING THE FOLK FROM THE THEOREM

Robustness in the presence of component damage is a property that is commonly attributed to neural networks. The content of the following statement embodies this sentiment.

*Folk Theorem 1: Computation in neural networks is not substantially affected by damage to network components.*

While such a statement is manifestly not true in general—witness networks with "grandmother cells" where damage to the critical cells fatally impairs the computational ability of the network—there is anecdotal evidence in support of it in

situations where the network has a more "distributed" flavour with relatively dense interconnectivity of elements and a distributed format for the storage of information. Qualitatively, the phenomenon is akin to holographic modes of storing information where the distributed, non-localised format of information storage carries with it a measure of security against component damage.

The flip side to the robust folk theorem is the following observation, robustness notwithstanding:

> *Folk Theorem 2: Dense interconnectivity is a sine qua non for efficient usage of resources; in particular, sparser structures exhibit a degradation in computational capability.*

Again, disclaimers have to be thrown in on the applicability of such a statement. In recurrent network architectures, however, this might seem to have some merit. In particular, in associative memory applications, while structural robustness might guarantee that the loss in memory storage capacity with increased interconnection sparsity may not be catastrophic, nonetheless intuitively a drop in capacity with increased sparsity may be expected.

This communication represents an effort to mathematically codify these tenets. In the setting we examine we formally introduce sparse network interconnectivity by invoking the notion of a (puckish) devil in the network which severs interconnection links between neurons. Our results here involve some surprising consequences—viewed in the light of the two folk theorems—of sparse interconnectivity to robustness and to memory storage capability. Only the main results are stated here; for extensions and details of proofs we refer the interested reader to Venkatesh (1990) and Biswas and Venkatesh (1990).

**Notation**   We denote by $\mathbb{B}$ the set $\{-1, 1\}$. For every integer $k$ we denote the set of integers $\{1, 2, \ldots, k\}$ by $[k]$. By ordered multiset we mean an ordered collection of elements with repetition of elements allowed, and by $k$-set we mean an ordered multiset of $k$ elements. All logarithms in the exposition are to base $e$.

## 2   RECURRENT NETWORKS

### 2.1   INTERCONNECTION GRAPHS

We consider a recurrent network of $n$ formal neurons. The allowed pattern of neural interconnectivity is specified by the edges of a *(bipartite) interconnectivity graph*, $G_n$, on vertices, $[n] \times [n]$. In particular, the existence of an edge $\{i, j\}$ in $G_n$ is indicative that the state of neuron $j$ is input to neuron $i$.[2] The network is characterised by an $n \times n$ matrix of weights, $\mathbf{W} = [w_{ij}]$, where $w_{ij}$ denotes the (real) weight modulating the state of neuron $j$ at the input of neuron $i$. If $\mathbf{u} \in \mathbb{B}^n$ is the current state of the system, an update, $u_i \mapsto u_i'$ of the state of neuron $i$ is

specified by the linear threshold rule

$$u_i' = \text{sgn}\left(\sum_{j:\{i,j\}\in G} w_{ij}u_j\right).$$

The network dynamics describe trajectories in a state space comprised of the vertices of the $n$-cube.[3] We are interested in an associative memory application where we wish to store a desired set of states—the *memories*—as fixed points of the network, and with the property that errors in an input representation of a memory are corrected and the memory retrieved.

## 2.2 DOMINATORS

Let $\mathbf{u} \in \mathbb{B}^n$ be a memory and $0 \le \rho < 1$ a parameter. Corresponding to the memory $\mathbf{u}$ we generate a probe $\hat{\mathbf{u}} \in \mathbb{B}^n$ by independently specifying the components, $\hat{u}_j$, of the probe as follows:

$$\hat{u}_j = \begin{cases} u_j & \text{with probability } 1 - \rho \\ -u_j & \text{with probability } \rho. \end{cases} \tag{1}$$

We call $\hat{\mathbf{u}}$ a *random probe with parameter* $\rho$.

**Definition 2.1** We say that a memory, $\mathbf{u}$, *dominates over a radius* $\rho n$ if, with probability approaching one as $n \to \infty$, the network corrects all errors in a random probe with parameter $\rho$ in one synchronous step. We call $\rho$ the *(fractional) dominance radius*. We also say that $\mathbf{u}$ is *stable* if it is a 0-dominator.

REMARKS: Note that stable memories are just fixed points of the network. Also, the expected number of errors in a probe is $\rho n$.

## 2.3 CODES

For given integers $m \ge 1$, $n \ge 1$, a *code*, $\mathcal{K}_n^m$, is a collection of ordered multisets of size $m$ from $\mathbb{B}^n$. We say that an $m$-set of memories is *admissible* iff it is in $\mathcal{K}_n^m$.[4] Thus, a code just specifies which $m$-sets are allowable as memories. Examples of codes include: the set of all multisets of size $m$ from $\mathbb{B}^n$; a single multiset of size $m$ from $\mathbb{B}^n$; all collections of $m$ mutually orthogonal vectors in $\mathbb{B}^n$; all $m$-sets of vectors in $\mathbb{B}^n$ in general position.

Define two ordered multisets of memories to be *equivalent* if they are permutations of one another. We define the *size* of a code, $\mathcal{K}_n^m$, to be the number of distinct equivalence classes of $m$-sets of memories. We will be interested in codes of relatively large size: $\log|\mathcal{K}_n^m|/n \to \infty$ as $n \to \infty$. In particular, we require at least an exponential number of choices of (equivalence classes of) admissible $m$-sets of memories.

## 2.4    CAPACITY

For each fixed $n$ and interconnectivity graph, $G_n$, an *algorithm*, $\mathcal{X}$, is a prescription which, given an $m$-set of memories, produces a corresponding set of interconnection weights, $w_{ij}$, $i \in [n]$, $\{i, j\} \in G_n$. For $m \geq 1$ let $\mathcal{A}(\mathbf{u}^1, \dots, \mathbf{u}^m)$ be some attribute of $m$-sets of memories. (The following, for instance, are examples of attributes of admissible sets of memories: all the memories are stable in the network generated by $\mathcal{X}$; almost all the memories dominate over a radius $\rho n$.) *For given $n$ and $m$, we choose a random $m$-set of memories, $\mathbf{u}^1, \dots, \mathbf{u}^m$, from the uniform distribution on $\mathcal{K}_n^m$.*

**Definition 2.2** Given interconnectivity graphs $G_n$, codes $\mathcal{K}_n^m$, and algorithm $\mathcal{X}$, a sequence, $\{C_n\}_{n=1}^{\infty}$, is a *capacity function for the attribute $\mathcal{A}$* (or $\mathcal{A}$-*capacity* for short) if for $\lambda > 0$ arbitrarily small:

a)   $\mathbf{P}\{\mathcal{A}(\mathbf{u}^1, \dots, \mathbf{u}^m)\} \to 1$ as $n \to \infty$ whenever $m \leq (1 - \lambda)C_n$;

b)   $\mathbf{P}\{\mathcal{A}(\mathbf{u}^1, \dots, \mathbf{u}^m)\} \to 0$ as $n \to \infty$ whenever $m \geq (1 + \lambda)C_n$.

We also say that $C_n$ is a *lower $\mathcal{A}$-capacity* if property (a) holds, and that $C_n$ is an *upper $\mathcal{A}$-capacity* if property (b) holds.

For $m \geq 1$ let $\mathbf{u}^1, \dots, \mathbf{u}^m \in \mathbb{B}^n$ be an $m$-set of memories chosen from a code $\mathcal{K}_n^m$. The *outer-product algorithm* specifies the interconnection weights, $w_{ij}$, according to the following rule: for $i \in [n]$, $\{i, j\} \in G_n$,

$$w_{ij} = \sum_{\beta=1}^{m} u_i^{\beta} u_j^{\beta}. \tag{2}$$

In general, if the interconnectivity graph, $G_n$, is symmetric then, under a suitable mode of operation, there is a Liapunov function for the network specified by the outer-product algorithm. *Given graphs $G_n$, codes $\mathcal{K}_n^m$, and the outer-product algorithm, for fixed $0 \leq \rho < 1/2$ we are interested in the attribute $\mathcal{D}_\rho$ that each of the $m$ memories dominates over a radius $\rho n$.*

## 3    RANDOM GRAPHS

We investigate the effect of a random loss of neural interconnections in a recurrent network of $n$ neurons by considering a random bipartite interconnectivity graph $RG_n$ on vertices $[n] \times [n]$ with

$$\mathbf{P}\{\{i, j\} \in RG_n\} = p$$

for all $i \in [n]$, $j \in [n]$, and with these probabilities being mutually independent. The interconnection probability $p$ is called the *sparsity parameter* and may depend on $n$. The system described above is formally equivalent to beginning with a fully-interconnected network of neurons with specified interconnection weights $w_{ij}$, and then invoking a devil which randomly severs interconnection links, independently retaining each interconnection weight $w_{ij}$ with probability $p$, and severing it (replacing it with a zero weight) with probability $q = 1 - p$.

Let $C\mathcal{K}_n^m$ denote the *complete code* of all choices of ordered multisets of size $m$ from $\mathbb{B}^n$.

**Theorem 3.1** *Let $0 \le \rho < 1/2$ be a fixed dominance radius, and let the sparsity parameter $p$ satisfy $pn^2 \to \infty$ as $n \to \infty$. Then $(1 - 2\rho)^2 pn/2\log pn^2$ is a $\mathcal{D}_\rho$-capacity for random interconnectivity graphs $RG_n$, complete codes $C\mathcal{K}_n^m$, and the outer-product algorithm.*

REMARKS:   The above result graphically validates Folk Theorem 1 on the fault-tolerant nature of the network; specifically, the network exhibits a *graceful degradation* in storage capacity as the loss in interconnections increases. Catastrophic failure occurs only when $p$ is smaller than $\log n/n$: *each neuron need retain only of the order of $\Omega(\log n)$ links of a total of $n$ possible links with other neurons for useful associative properties to emerge.*

## 4   BLOCK GRAPHS

One of the simplest (and most regular) forms of sparsity that a favourably disposed devil might enjoin is block sparsity where the neurons are partitioned into disjoint subsets of neurons with full-interconnectivity within each subset and no neural interconnections between subsets. The weight matrix in this case takes on a block diagonal form, and the interconnectivity graph is composed of a set of disjoint, complete bipartite sub-graphs.

More formally, let $1 \le b \le n$ be a positive integer, and let $\{I_1, \ldots, I_{n/b}\}$ partition $[n]$ such that each subset of indices, $I_k$, $k = 1, \ldots, n/b$, has size $|I_k| = b$.[5] We call each $I_k$ a *block* and $b$ the *block size*. We specify the edges of the *(bipartite) block interconnectivity graph* $BG_n$ by $\{i, j\} \in BG_n$ iff $i$ and $j$ lie in a common block.

**Theorem 4.1** *Let the block size $b$ be such that $b = \Omega(n)$ as $n \to \infty$, and let $0 \le \rho < 1/2$ be a fixed dominance radius. Then $(1-2\rho)^2 b/2\log bn$ is a $\mathcal{D}_\rho$-capacity for block interconnectivity graphs $BG_n$, complete codes $C\mathcal{K}_n^m$, and the outer-product algorithm.*

**Corollary 4.2** *Under the conditions of theorem 4.1 the fixed point memory capacity is $b/2\log bn$.*

**Corollary 4.3** *For a fully-interconnected graph, complete codes $C\mathcal{K}_n^m$, and the outer-product algorithm, the fixed point memory capacity is $n/4\log n$.*

Corollary 4.3 is the main result shown by McEliece, Posner, Rodemich, and Venkatesh (1987). Theorem 4.1 extends the result and shows (formally validating the intuition espoused in Folk Theorem 2) that increased sparsity causes a loss in capacity if the code is complete, i.e., all choices of memories are considered admissible. It is possible, however, to design codes to take advantage of the sparse interconnectivity structure, rather at odds with the Folk Theorem.

Without loss of generality let us assume that block $I_1$ consists of the first $b$ indices, $[b]$, block $I_2$ the next $b$ indices, $[2b]-[b]$, and so on, with the last block $I_{n/b}$ consisting of the last $b$ indices, $[n]-[n-b]$. We can then partition any vector $\mathbf{u} \in \mathbb{B}^n$ as

$$\mathbf{u} = \begin{pmatrix} \mathbf{u}_1 \\ \mathbf{u}_2 \\ \vdots \\ \mathbf{u}_{n/b} \end{pmatrix}, \qquad (3)$$

where for $k = 1, \ldots, n/b$, $\mathbf{u}_k$ is the vector of components corresponding to block $I_k$. For $M \geq 1$ we form the *block code* $\mathcal{BK}_n^{M^{n/b}}$ as follows: to each ordered multiset of $M$ vectors, $\mathbf{u}^1, \ldots, \mathbf{u}^M$ from $\mathbb{B}^n$, we associate a unique ordered multiset in $\mathcal{BK}_n^{M^{n/b}}$ by lexicographically ordering all $M^{n/b}$ vectors of the form

$$\begin{pmatrix} \mathbf{u}_1^{\alpha_1} \\ \mathbf{u}_2^{\alpha_2} \\ \vdots \\ \mathbf{u}_{n/b}^{\alpha_{n/b}} \end{pmatrix}, \qquad \alpha_1, \alpha_2, \ldots, \alpha_{n/b} \in [M].$$

Thus, we obtain an admissible set of $M^{n/b}$ memories from any ordered multiset of $M$ vectors in $\mathbb{B}^n$ by "mixing" the blocks of the vectors. We call each $M$-set of vectors, $\mathbf{u}^1, \ldots, \mathbf{u}^M \in \mathbb{B}^n$, the *generating vectors* for the corresponding admissible set of memories in $\mathcal{BK}_n^{M^{n/b}}$.

EXAMPLE: Consider a case with $n = 4$, block size $b = 2$, and $M = 2$ generating vectors. To any 2-set of generating vectors there corresponds a unique $4(=M^{n/b})$-set in the block code as follows:

$$\begin{pmatrix} u_1^1 \\ u_2^1 \\ \hline u_3^1 \\ u_4^1 \end{pmatrix}, \begin{pmatrix} u_1^2 \\ u_2^2 \\ \hline u_3^2 \\ u_4^2 \end{pmatrix} \longmapsto \begin{pmatrix} u_1^1 \\ u_2^1 \\ \hline u_3^1 \\ u_4^1 \end{pmatrix}, \begin{pmatrix} u_1^1 \\ u_2^1 \\ \hline u_3^2 \\ u_4^2 \end{pmatrix}, \begin{pmatrix} u_1^2 \\ u_2^2 \\ \hline u_3^1 \\ u_4^1 \end{pmatrix}, \begin{pmatrix} u_1^2 \\ u_2^2 \\ \hline u_3^2 \\ u_4^2 \end{pmatrix}.$$

**Theorem 4.4** *Let $0 \leq \rho < 1/2$ be a fixed dominance radius. Then we have the following capacity estimates for block interconnectivity graphs $BG_n$, block codes $\mathcal{BK}_n^m$, and the outer-product algorithm:*

a) *If the block size $b$ satisfies $n \log \log bn / b \log bn \to 0$ as $n \to \infty$ then the $\mathcal{D}_\rho$-capacity is*

$$\left[ \frac{(1-2\rho)^2 b}{2 \log bn} \right]^{n/b}.$$

b) *Define for any $\nu$*

$$C_n(\nu) = 2^{\frac{n \log b}{b \log 2} \left[ 1 - \frac{\log \log bn + \log\left( 2(1-2\rho)^{-2} \right)}{\log b} + \frac{\nu \log \log bn}{(\log b)(\log bn)} \right]}.$$

*If the block size b satisfies $b/\log n \to \infty$ and $b \log bn/\log \log bn = O(n)$ as $n \to \infty$, then $C_n(\nu)$ is a lower $\mathcal{D}_\rho$-capacity for any choice of $\nu < 3/2$ and $C_n(\nu)$ is an upper $\mathcal{D}_\rho$-capacity for any $\nu > 3/2$.*

**Corollary 4.5** *If, for fixed $t \geq 1$, we have $b = n/t$, then, under the conditions of theorem 4.4, the $\mathcal{D}_\rho$-capacity is*

$$(1 - 2\rho)^{2t} t^{-t} 4^{-t} \left(\frac{n}{\log n}\right)^t.$$

**Corollary 4.6** *For any fixed dominance radius $0 \leq \rho < 1/2$, and for any $\tau < 1$, a constant $c > 0$ and a code of size $\Omega\left(2^{cn^{2-r}}\right)$ can be found such that it is possible to achieve lower $\mathcal{D}_\rho$-capacities which are $\Omega\left(2^{n^r}\right)$ in recurrent neural networks with interconnectivity graphs of degree $\Theta\left(n^{1-\tau}\right)$.*

REMARKS:   If the number of blocks is kept fixed as $n$ grows (i.e., the block size grows linearly with $n$) then capacities polynomial in $n$ are attained. If the number of blocks increases with $n$ (i.e., the block size grows sub-linearly with $n$) then super-polynomial capacities are attained. Furthermore, we have the surprising result rather at odds with Folk Theorem 2 that very large storage capacities can be obtained at the expense of code size (while still retaining large code sizes) in *increasingly sparse networks.*

## Acknowledgements

The support of research grants from E. I. Dupont de Nemours, Inc. and the Air Force Office of Scientific Research (grant number AFOSR 89–0523) is gratefully acknowledged.

## Footnotes

[1]Well, maybe an imp.

[2]Equivalently, imagine a devil loose with a pair of scissors snipping those interconnections for which $\{i, j\} \notin G_n$. For a complementary discussion of sparse interconnectivity see Komlós and Paturi (1988).

[3] As usual, there are Liapunov functions for the system under suitable conditions on the interconnectivity graph and the corresponding weights.

[4] We define admissible $m$-sets of memories in terms of ordered multisets rather than sets so as to obviate certain technical nuisances.

[5] Here, as in the rest of the paper, we ignore details with regard to integer rounding.

## References

Biswas, S. and S. S. Venkatesh (1990), "Codes, sparsity, and capacity in neural associative memory," submitted for publication.

Komlós, J. and R. Paturi (1988), "Effects of connectivity in associative memory models," Technical Report CS88–131, University of California, San Diego, 1988.

McEliece, R. J., E. C. Posner, E. R. Rodemich, and S. S. Venkatesh (1987), "The capacity of the Hopfield associative memory," *IEEE Trans. Inform. Theory*, vol. IT–33, pp. 461–482.

Venkatesh, S. S. (1990), "Robustness in neural computation: random graphs and sparsity," to appear *IEEE Trans. Inform. Theory.*